# SIMULATION AND MEASUREMENT OF THE ELECTRIC FIELDS GENERATED BY WEAKLY ELECTRIC FISH

Brian Rasnow[1], Christopher Assad[2], Mark E. Nelson[3] and James M. Bower[3]

Divisions of Physics[1], Electrical Engineering[2], and Biology[3]

Caltech, Pasadena, 91125

## ABSTRACT

The weakly electric fish, *Gnathonemus petersii*, explores its environment by generating pulsed electric fields and detecting small perturbations in the fields resulting from nearby objects. Accordingly, the fish detects and discriminates objects on the basis of a sequence of electric "images" whose temporal and spatial properties depend on the timing of the fish's electric organ discharge and its body position relative to objects in its environment. We are interested in investigating how these fish utilize timing and body-position during exploration to aid in object discrimination. We have developed a finite-element simulation of the fish's self-generated electric fields so as to reconstruct the electrosensory consequences of body position and electric organ discharge timing in the fish. This paper describes this finite-element simulation system and presents preliminary electric field measurements which are being used to tune the simulation.

## INTRODUCTION

The active positioning of sensory structures (i.e. eyes, ears, whiskers, nostrils, etc.) is characteristic of the information seeking behavior of all exploratory animals. Yet, in most existing computational models and in many standard experimental paradigms, the active aspects of sensory processing are either eliminated or controlled (e.g. by stimulating fixed groups of receptors or by stabilizing images). However, it is clear that the active positioning of receptor surfaces directly affects the content and quality of the sensory information received by the nervous system. Thus, controlling the position of sensors during sensory exploration constitutes an important feature of an animals strategy for making sensory discriminations. Quantitative study of this process could very well shed light on the algorithms and internal representations used by the nervous system in discriminating peripheral objects.

Studies of the active use of sensory surfaces generally can be expected to pose a number of experimental challenges. This is because, in many animals, the sensory surfaces involved are themselves structurally complicated, making it difficult to reconstruct position sequences or the consequences of any repositioning. For example, while the sen-

sory systems of rats have been the subjects of a great deal of behavioral (Welker, 1964) and neurophysiological study (Gibson & Welker, 1983), it is extremely difficult to even monitor the movements of the perioral surfaces (lips, snout, whiskers) used by these animals in their exploration of the world let alone reconstruct the sensory consequences. For these reasons we have sought an experimental animal with a sensory system in which these sensory-motor interactions can be more readily quantified.

The experimental animal which we have selected for studying the control of sensory surface position during exploration is a member of a family of African freshwater fish (Mormiridae) that use self-generated electric fields to detect and discriminate objects in their environment (Bullock & Heiligenberg, 1986). The electrosensory system in these fish relies on an "electric organ" in their tails which produces a weak pulsed electric field in the surrounding environment (significant within 1-2 body lengths) that is then detected with an array of electrosensors that are extremely sensitive to voltage drops across the skin. These "electroreceptors" allow the fish to respond to the perturbations in the electric field resulting from objects in the environment which differ in conductivity from the surrounding water (Fig. 1).

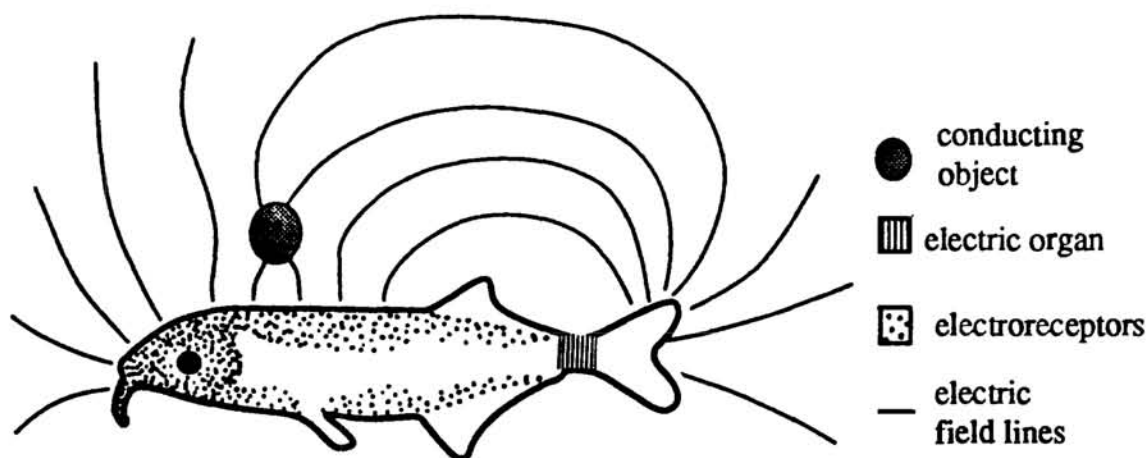

conducting
object

electric organ

electroreceptors

electric
field lines

Figure 1.  The peripheral electrosensory system of *Gnathonemus petersii* consists of an "electric organ" current source at the base of the tail and several thousand "electroreceptor" cells distributed nonuniformly over the fish's body.  A conducting object near the fish causes a local increase in the current through the skin.

These fish are nocturnal, and rely more on their electric sense than on any other sensory system in performing a wide range of behaviors (eg. detecting and localizing objects such as food). It is also known that these fish execute exploratory movements, changing their body position actively as they attempt an electrosensory discrimination (Toerring & Belbenoit, 1979). Our objective is to understand how these movements change the distribution of the electric field on the animals skin, and to determine what, if any, relationship this has to the discrimination process.

There are several clear advantages of this system for our studies. First, the electrore-

ceptors are in a fixed position with respect to each other on the surface of the animal. Therefore, by knowing the overall body position of the animal it is possible to know the exact spatial relationship of electroreceptors with respect to objects in the environment. Second, the physical equations governing the self-generated electric field in the fish's environment are well understood. As a consequence, it is relatively straightforward to reconstruct perturbations in the electric field resulting from objects of different shape and conductance. Third, the electric potential can be readily measured, providing a direct measure of the electric field at a distance from the fish which can be used to reconstruct the potential difference across the animals skin. And finally, in the particular species of fish we have chosen to work with, *Gnathonemus petersii*, individual animals execute a brief (100 µsec) electric organ discharge (EOD) at intervals of 30 msec to a few seconds. Modification of the firing pattern is known to be correlated with changes in the electrical environment (Lissmann, 1958). Thus, when the electric organ discharges, it is probable that the animal is interested in "taking a look" at its surroundings. In few other sensory systems is there as direct an indication of the attentional state of the subject.

Having stated the advantages of this system for the study we have undertaken, it is also the case that considerable effort will still be necessary to answer the questions we have posed. For example, as described in this paper, in order to use electric field measurements made at a distance to infer the voltages across the surface of the animal's skin, it is necessary to develop a computer model of the fish and its environment. This will allow us to predict the field on the animal's skin surface given different body positions relative to objects in the environment. This paper describes our first steps in constructing this simulation system.

## Experimental Approach and Methods

### Simulations of Fish Electric Fields

The electric potential, $\Phi(x)$, generated by the EOD of a weakly electric fish in a fish tank is a solution of Poisson's equation:

$$\nabla \cdot (\rho \nabla \Phi) = f$$

where $\rho(x)$ and $f(x)$ are the impedance magnitude and source density at each point x inside and surrounding the fish. Our goal is to solve this equation for $\Phi$ given the current source density, f, generated by the electric organ and the impedances, $\rho$, corresponding to the properties of the fish and external objects (rocks, worms, etc.). Given $\rho$ and f, this equation can be solved for the potential $\Phi$ using a variety of iterative approximation schemes. Iterative methods, in general, first discretize the spatial domain of the problem into a set of "node" points, and convert Poisson's equation into a set of algebraic equations with the nodal potentials as the unknown parameters. The node values, in this case, each represent an independent degree of freedom of the system and, as a consequence, there are as many equations as there are nodes. This very large system of equations can

then be solved using a variety of standard techniques, including relaxation methods, conjugate gradient minimization, domain decomposition and multi-grid methods.

To simulate the electric fields generated by a fish, we currently use a 2-dimensional finite element domain discretization (Hughes, 1987) and conjugate gradient solver. We chose the finite element method because it allows us to simulate the electric fields at much higher resolution in the area of interest close to the animal's body where the electric field is largest and where errors due to the discretization would be most severe. The finite element method is based on minimizing a global function that corresponds to the potential energy of the electric field. To compute this energy, the domain is decomposed into a large number of elements, each with uniform impedance (see Fig. 2). The global energy is expressed as a sum of the contributions from each element, where the potential within each element is assumed to be a linear interpolation of the potentials at the nodes or vertices of each element. The conjugate gradient solver determines the values of the node potentials which minimize the global energy function.

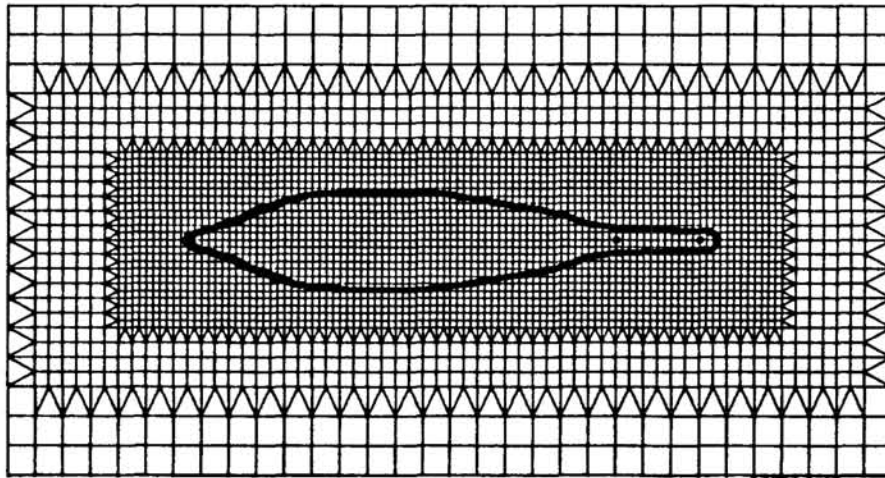

**Figure 2.** The inner region of a finite element grid constructed for simulating in 2-dimensions the electric field generated by an electric fish.

## Measurement of Fish Electric Fields

Another aspect of our experimental approach involves the direct measurement of the potential generated by a fish's EOD in a fish tank using arrays of small electrodes and differential amplifiers. The electrodes and electronics have a high impedance which minimizes their influence on the electric fields they are designed to measure. The electrodes are made by pulling a 1mm glass capillary tube across a heated tungsten filament, resulting in a fine tapered tip through which a 100μm silver wire is run. The end of this wire is melted in a flame leaving a 200μm ball below the glass insulation. Several electrodes are then mounted as an array on a microdrive attached to a modified X-Y plotter under computer control and giving better than 1mm positioning accuracy. Generated potentials are amplified by a factor of 10 - 100, and digitized at a rate of 100kHz per channel with a 12 bit A/D converter using a Masscomp 5700 computer. An array processor searches this

continuous stream of data for EOD waveforms, which are extracted and saved along with the position of the electrode array.

## Results

## Calibration of the Simulator

In order to have confidence in the overall system, it was first necessary to calibrate both the recording and the simulation procedures. To do this we set up relatively simple geometrical arrangements of sources and conductors in a fish tank for which the potential could be found analytically. The calibration source was an electronic "fake fish" circuit that generated signals resembling the discharge of *Gnathonemus*.

## Point current source

A point source in a 2-dimensional box is perhaps the simplest configuration with which to initially test our electric field reconstruction system. The analytic solution for the potential from a point current source centered in a grounded, conducting 2-dimensional box is:

$$\Phi(x, y) = \sum_{n=1}^{\infty} \frac{\sin\left(\frac{n\pi}{2}\right)\sin\left(\frac{n\pi x}{L}\right)\sinh\left(\frac{n\pi y}{L}\right)}{n\, L\, \cosh\left(\frac{n\pi}{2}\right)}$$

Our finite element simulation, based on a regular 80 x 80 node grid differs from the above expression by less than 1%, except in the elements adjacent to the source, where the potential change across these elements is large and is not as accurately reconstructed by a linear interpolation (Fig. 3). Smaller elements surrounding the source would improve the accuracy, however, one should note the analytic solution is infinite at the location of the "point" source whereas the measured and simulated sources (and real fish) have finite current densities.

To measure the real equivalent of a point source in a 2-dimensional box, we used a linear current source (a wire) which ran the full depth of a real 3-dimensional tank. Measurements made in the midplane of the tank agree with the simulation and analytic solution to better than 5% (Fig. 3.). Uncertainty in the positions of the current source and recording sites relative to the position of the conducting walls probably accounts for much of this difference.

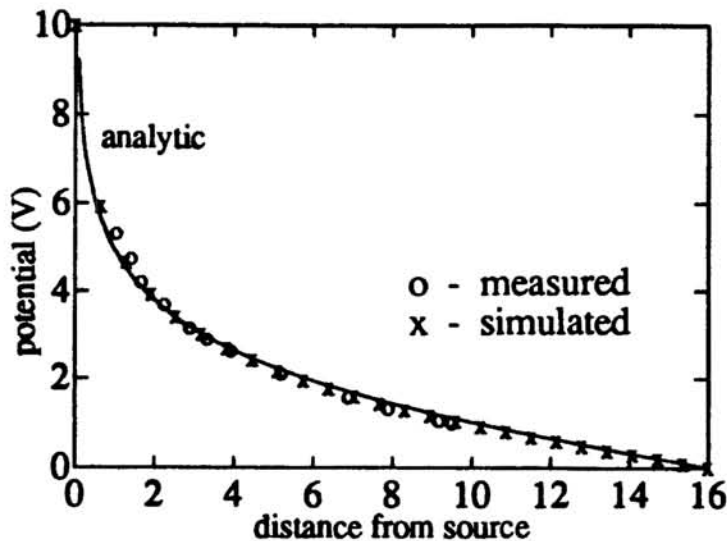

**Figure 3.** Electric potential of a point current source centered in a grounded 2-dimensional box.

## Measurements of Fish Fields and 2-Dimensional Simulations

Calibration of our finite element model of an electric fish requires direct measurements of the electric potential close to a discharging fish. Fig. 4 shows a recording of a single EOD sampled with 5 colinear electrodes near a restrained fish. The waveform is bipolar, with the first phase positive if recorded near the animals head and negative if recorded near the tail (relative to a remote reference). We used the peak amplitude of the larger second phase of the waveform to quantify the electric potential recorded at each location. Note that the potential reverses sign at a point approximately midway along the tail. This location corresponds to the location of the null potential shown in Fig. 5.

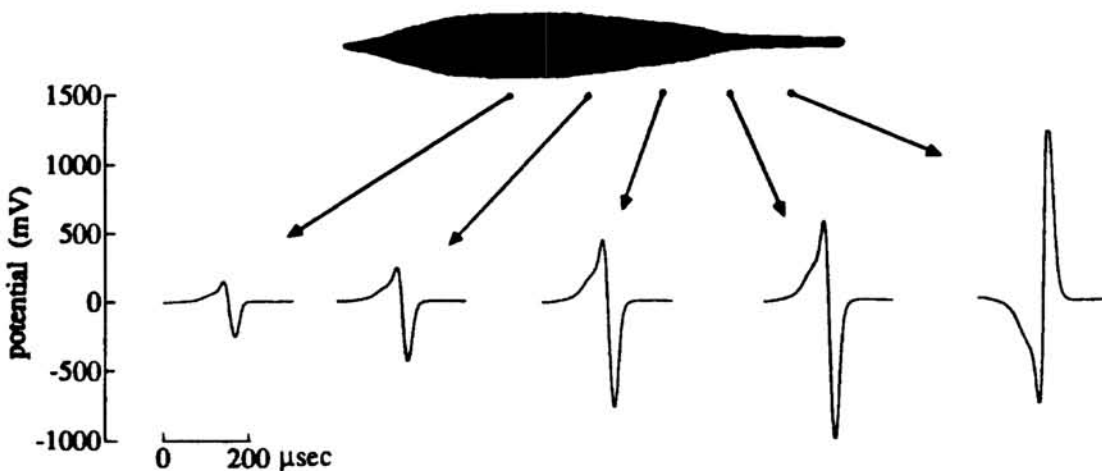

Figure 4. EOD waveform sampled simultaneously from 5 electrodes.

Measurements of EODs from a restrained fish exhibited an extraordinarily small variance in amplitude and waveform over long periods of time. In fact, the peak-peak amplitude of the EOD varied by less than 0.4% in a sample of 40 EOD's randomly chosen during a 30 minute period. Thus we are able to directly compare waveforms sampled sequentially without renormalizing for fluctuations in EOD amplitude.

Figure 5 shows equipotential lines reconstructed from a set of 360 measurements made in the midplane of a restrained *Gnathonemus*. Although the observed potential resembles that from a purely dipolar source (Fig. 6), careful inspection reveals an asymmetry between the head and tail of the fish. This asymmetry can be reproduced in our simulations by adjusting the electrical properties of the fish. Qualitatively, the measured fields can be reproduced by assigning a low impedance to the internal body cavity and a high impedance to the skin. However, in order to match the location of the null potential, the skin impedance must vary over the length of the body. We are currently quantifying these parameters, as described in the next section.

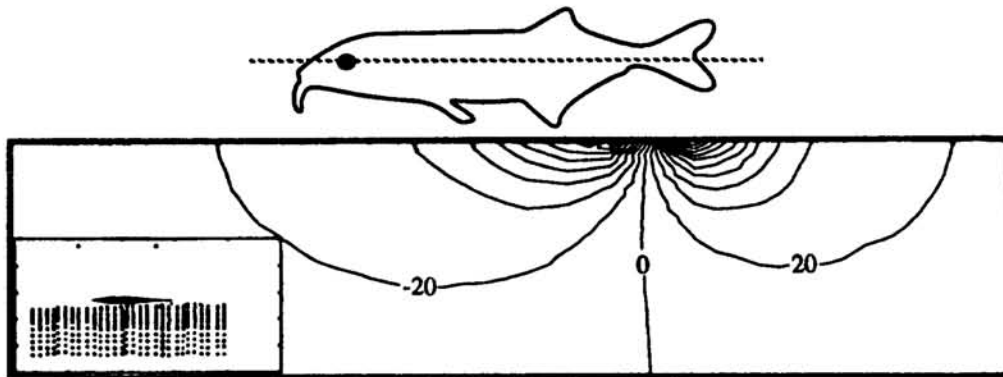

**Figure 5.** Measured potentials (at peak of second phase of EOD) recorded from a restrained *Gnathonemus petersii* in the midplane of the fish. Equipotential lines are 20 mV apart. Inset shows relative location of fish and sampling points in the fish tank.

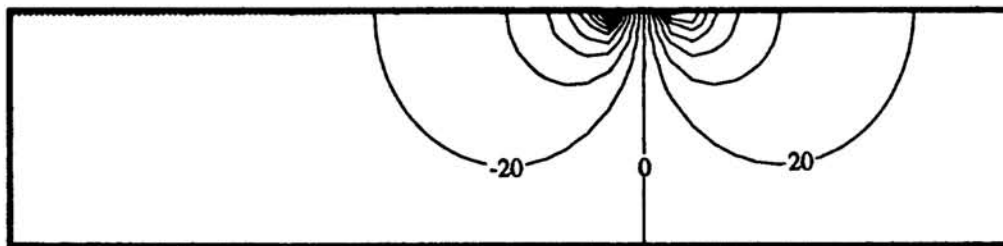

**Figure 6.** Equipotential lines from a 2-dimensional finite element simulation of a dipole using the grid of Fig. 2. The resistivity of the fish was set equal to that of the surroundings in this simulation.

## Future Directions

There is still a substantial amount of work that remains to be done before we achieve our goal of being able to fully reconstruct the pattern of electroreceptor activation for any arbitrary body position in any particular environment. First, it is clear that we require more information about the electrical structure of the fish itself. We need an accurate representation of the internal impedance distribution $\rho(x)$ of the body and skin as well as of the source density $f(x)$ of the electric organ. To some extent this can be addressed as an inverse problem, namely given the measured potential $\Phi(x)$, what choice of $\rho(x)$ and $f(x)$ best reproduces the data. Unfortunately, in the absence of further constraints, there are many equally valid solution, thus we will need to directly measure the skin and body impedance of the fish. Second, we need to extend our finite-element simulations of the fish to 3-dimensions which, although conceptually straight forward, requires substantial technical developments to be able to (a) specify and visualize the space-filling set of 3-dimensional finite-elements (eg. tetrahedrons) for arbitrary configurations, (b) compute the solution to the much larger set of equations (typically a factor of 100-1000) in a reasonable time, and (c) visualize and analyze the resulting solutions for the 3-dimensional electrical fields. As a possible solution to (b), we are developing and testing a parallel processor implementation of the simulator.

## References

Bullock, T. H. & Heiligenberg, W. (Eds.) (1986). "Electroreception", Wiley & Sons, New York.

Gibson, J. M. & Welker, W. I. (1983). Quantitative Studies of Stimulus Coding in First-Order Vibrissa Afferents of Rats. 1. Receptive Field Properties and Threshold Distributions. *Somatosensory Res.* 1:51-67.

Hughes, T. J. (1987). The Finite Element Method: Linear Static and Dynamic Finite Element Analysis. Prentice-Hall, New Jersey.

Lissmann, H.W. (1958). On the function and evolution of electric organs in fish. *J. Exp. Biol.* 35:156-191.

Toerring, M. J. and Belbenoit, P. (1979). Motor Programmes and Electroreception in Mormyrid Fish. *Behav. Ecol. Sociobiol.* 4:369-379.

Welker, W. I. (1964). Analysis of Sniffing of the Albino Rat. *Behaviour* 22:223-244.
